# Hippocampal Model of Rat Spatial Abilities Using Temporal Difference Learning

**David J Foster***
Centre for Neuroscience
Edinburgh University

**Richard GM Morris**
Centre for Neuroscience
Edinburgh University

**Peter Dayan**
E25-210, MIT
Cambridge, MA 02139

## Abstract

We provide a model of the standard watermaze task, and of a more challenging task involving novel platform locations, in which rats exhibit one-trial learning after a few days of training. The model uses hippocampal place cells to support reinforcement learning, and also, in an integrated manner, to build and use allocentric *coordinates*.

## 1 INTRODUCTION

Whilst it has long been known both that the hippocampus of the rat is needed for normal performance on spatial tasks[13, 11] and that certain cells in the hippocampus exhibit place-related firing,[12] it has not been clear how place cells are actually used for navigation. One of the principal conceptual problems has been understanding how the hippocampus could specify or learn paths to goals when spatially tuned cells in the hippocampus respond only on the basis of the rat's current location. This work uses recent ideas from reinforcement learning to solve this problem in the context of two rodent spatial learning results.

Reference memory in the watermaze[11] (RMW) has been a key task demonstrating the importance of the hippocampus for spatial learning. On each trial, the rat is placed in a circular pool of cloudy water, the only escape from which is a platform which is hidden (below the water surface) but which remains in a constant position. A random choice of starting position is used for each trial. Rats take asymptotically short paths after approximately 10 trials (see figure 1 a). Delayed match-to-place (DMP) learning is a refined version in which the platform's location is changed on each day. Figure 1b shows escape latencies for rats given four trials per day for nine days, with the platform in a novel position on each day. On early days, acquisition

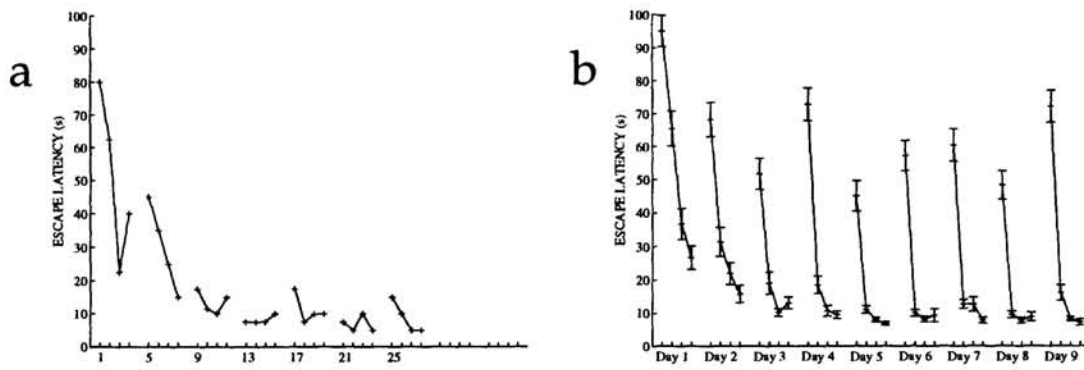

Figure 1: a) Latencies for rats on the reference memory in the watermaze (RMW) task (N=8). b) Latencies for rats on the Delayed Match-to-Place (DMP) task (N=62).

is gradual but on later days, rats show one-trial learning, that is, near asymptotic performance on the second trial to a novel platform position.

The RMW task has been extensively modelled.[6,4,5,20] By contrast, the DMP task is new and computationally more challenging. It is solved here by integrating a standard actor-critic reinforcement learning system[2,7] which guarantees that the rat will be competent to perform well in arbitrary mazes, with a system that learns spatial *coordinates* in the maze. Temporal difference learning[17] (TD) is used for actor, critic *and* coordinate learning. TD learning is attractive because of its generality for arbitrary Markov decision problems and the fact that reward systems in vertebrates appear to instantiate it.[14]

## 2   THE MODEL

The model comprises two distinct networks (figure 2): the actor-critic network and a coordinate learning network. The contribution of the hippocampus, for both networks, is to provide a state-space representation in the form of place cell basis functions. Note that only the activities of place cells are required, by contrast with decoding schemes which require detailed information about each place cell.[4]

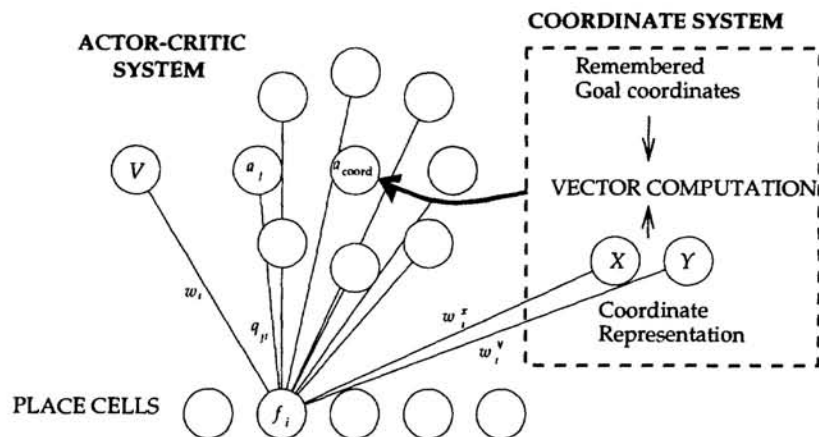

Figure 2: Model diagram showing the interaction between actor-critic and coordinate system components.

## 2.1 Actor-Critic Learning

Place cells are modelled as being tuned to location. At position **p**, place cell $i$ has an output given by $f_i(\mathbf{p}) = \exp\{-\|\mathbf{p} - \mathbf{s}_i\|^2/2\sigma^2\}$, where $\mathbf{s}_i$ is the place field centre, and $\sigma = 0.1$ for all place fields. The critic learns a value function $\hat{V}(\mathbf{p}) = \sum_i w_i f_i(\mathbf{p})$ which comes to represent the distance of **p** from the goal, using the TD rule $\Delta w_i^t \propto \delta^t f_i(\mathbf{p}^t)$, where

$$\delta^t = r(\mathbf{p}^t, \mathbf{p}^{t+1}) + \gamma \hat{V}(\mathbf{p}^{t+1}) - \hat{V}(\mathbf{p}^t) \qquad (1)$$

is the TD error, $\mathbf{p}^t$ is position at time $t$, and the reward $r(\mathbf{p}^t, \mathbf{p}^{t+1})$ is 1 for any move onto the platform, and 0 otherwise. In a slight alteration of the original rule, the value $V(\mathbf{p})$ is set to zero when **p** is *at* the goal, thus ensuring that the total future rewards for moving onto the goal will be exactly 1. Such a modification improves stability in the case of TD learning with overlapping basis functions. The discount factor, $\gamma$, was set to 0.99. Simultaneously the rat refines a policy, which is represented by eight action cells. Each action cell ($a_j$ in figure 2) receives a parameterised input at any position **p**: $a_j(\mathbf{p}) = \sum_i q_{ji} f_i(\mathbf{p})$. An action is chosen stochastically with probabilities given by $P(a_j) = \exp\{2a_j\}/\sum_k \exp\{2a_k\}$. Action weights are reinforced according to:[2]

$$\Delta q_{ji}^t \propto \delta^t f_i(\mathbf{p}^t) g_j(\theta^t) \qquad (2)$$

where $g_j(\theta^t)$ is a gaussian function of the difference between the head direction $\theta^t$ at time $t$ and the preferred direction of the $j$th action cell. Figure 3 shows the development of a policy over a few trials.

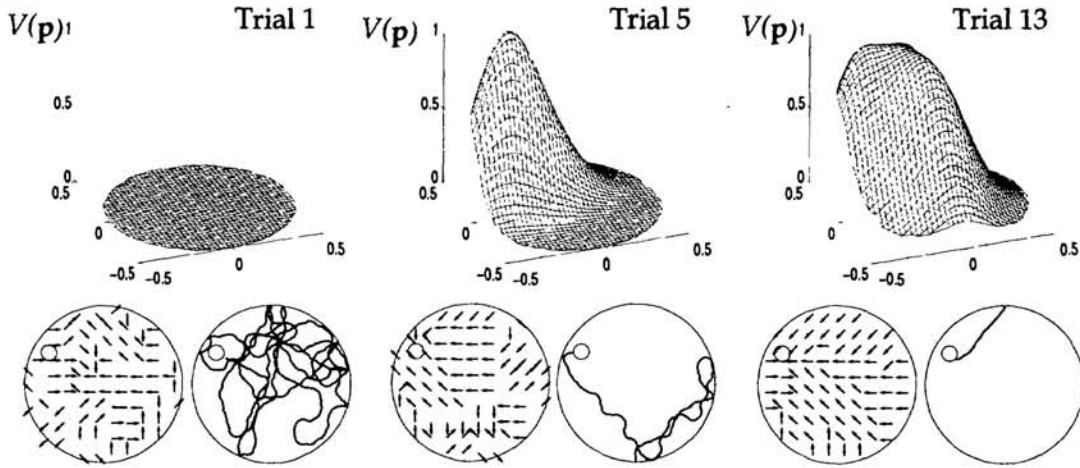

Figure 3: The RMW task: the value function gradually disseminates information about reward proximity to all regions of the environment. Policies and paths are also shown.

There is no analytical guarantee for the convergence of TD learning with policy adaptation. However our simulations show that the algorithm always converges for the RMW task. In a simulated arena of diameter 1m and with swimming speeds of 20cm/s, the simulation matched the performance of the real rats very closely (see figure 5). This demonstrates that TD-based reinforcement learning is adequately fast to account for the learning performance of real animals.

## 2.2  Coordinate Learning

Although the learning of a value function and policy is appropriate for finding a fixed platform, the actor-critic model does not allow the transfer of knowledge from the task defined by one goal position to that defined by any other; thus it could not generate the sort of one-trial learning that is shown by rats on the DMP task (see figure 1b). This requires acquisition of some goal-independent knowledge about space. A natural mechanism for this is the path integration or self-motion system.[20,10] However, path integration presents two problems. First, since the rat is put into the maze in a different position for each trial, how can it learn *consistent* coordinates across the whole maze? Second, how can a general, powerful, but slow, behavioral learning mechanism such as TD be integrated with a specific, limited, but fast learning mechanism involving spatial coordinates?

Since TD critic learning is based on enforcing consistency in estimates of future reward, we can also use it to learn spatially consistent coordinates on the basis of samples of self-motion. It is assumed that the rat has an allocentric frame of reference.[18] The model learns parameterised estimates of the $x$ and $y$ coordinates of all positions $\mathbf{p}$: $x(\mathbf{p}) = \sum_i w_i^x f_i(\mathbf{p})$ and $y(\mathbf{p}) = \sum_i w_i^y f_i(\mathbf{p})$. Importantly, while place cells were again critical in supporting spatial representation, *they do not embody a map of space*. The coordinate functions, like the value function previously, have to be learned.

As the simulated rat moves around, the coordinate weights $\{w_i^x\}$ are adjusted according to:

$$\Delta w_i^x \propto \left( \Delta \hat{x}^t + \hat{X}(\mathbf{p}^{t+1}) - \hat{X}(\mathbf{p}^t) \right) \sum_{k=1}^t \lambda^{t-k} f_i(\mathbf{p}^k) \tag{3}$$

where $\Delta \hat{x}_t$ is the self-motion estimate in the $x$ direction. A similar update is applied to $\{w_i^y\}$. In this case, the full TD($\lambda$) algorithm was used (with $\lambda = 0.9$); however TD(0) could also have been used, taking slightly longer. Figure 4a shows the $x$ and $y$ coordinates at early and late phases of learning. It is apparent that they rapidly become quite accurate – this is an extremely easy task in an open field maze.

An important issue in the learning of coordinates is *drift*, since the coordinate system receives no direct information about the location of the origin. It turns out that the three controlling factors over the implicit origin are: the boundary of the arena, the prior setting of the coordinate weights (in this case all were zero) and the position and prior value of any absorbing area (in this case the platform). If the coordinate system as a whole were to drift once coordinates have been established, this would invalidate coordinates that have been remembered by the rat over long periods. However, since the expected value of the prediction error at time steps should be zero for any self-consistent coordinate mapping, such a mapping should remain stable. This is demonstrated for a single run: figure 4b shows the mean value of coordinates $x$ evolving over trials, with little drift after the first few trials.

We modeled the coordinate system as influencing the choice of swimming direction in the manner of an abstract action.[15] The (internally specified) coordinates of the most recent goal position are stored in short term memory and used, along with the current coordinates, to calculate a vector heading. This vector heading is thrown into the stochastic competition with the other possible actions, governed by a single weight which changes in a similar manner to the other action weights (as in equation 2, see also fig 4d), depending on the TD error, and on the angular proximity of the current head direction to the coordinate direction. Thus, whether the the coordinate-based direction is likely to be used depends upon its past performance.

One simplification in the model is the treatment of extinction. In the DMP task,

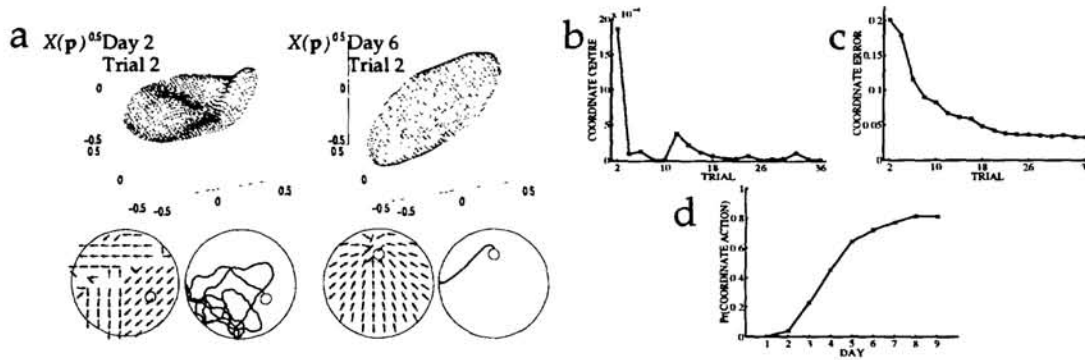

Figure 4: The evolution of the coordinate system for a typical simulation run: a.) coordinate outputs at early and late phases of learning, b.) the extent of drift in the coordinates, as shown by the mean coordinate value for a single run, c.) a measure of coordinate error for the same run $\hat{\sigma}_E^2 = \frac{\sum_r \sum_k \{\hat{X}_r(\mathbf{p}_k) - \bar{X}_r - X(\mathbf{p}_k)\}^2}{(N_p - 1)N_r}$, where $k$ indexes measurement points (max $N_p$) and $r$ indexes runs (max $N_r$), $X_r(\mathbf{p}_k)$ is the model estimate of $X$ at position $\mathbf{p}_k$, $X(\mathbf{p}_k)$ is the ideal estimate for a coordinate system centred on zero, and $\bar{X}_r$ is the mean value over all the model coordinates, d.) the increase during training of the probability of choosing the abstract action. This demonstrates the integration of the coordinates into the control system.

real rats extinguish to a platform that has moved fairly quickly whereas the actor-critic model extinguishes far more slowly. To get around this, when a simulated rat reaches a goal that has just been moved, the value and action weights are reinitialised, but the coordinate weights $w_i^x$ and $w_i^y$, and the weights for the abstract action, are not.

## 3   RESULTS

The main results of this paper are the replication by simulation of rat performance on the RMW and DMP tasks. Figures 1a and b show the course of learning for the rats; figures 5a and b for the model. For the DMP task, one-shot acquisition is apparent by the end of training.

## 4   DISCUSSION

We have built a model for one-trial spatial learning in the watermaze which uses a single TD learning algorithm in two separate systems. One system is based on a reinforcement learning that can solve general Markovian decision problems, and the other is based on coordinate learning and is specialised for an open-field water maze. Place cells in the hippocampus offer an excellent substrate for learning the actor, the critic and the coordinates.

The model is explicit about the relationship between the general and specific learning systems, and the learning behavior shows that they integrate seamlessly. As currently constituted, the coordinate system would fail if there were a barrier in the maze. We plan to extend the model to allow the coordinate system to specify abstract targets other than the most recent platform position – this could allow it fast navigation around a larger class of environments. It is also important to improve the model of learning 'set' behavior – the information about the nature of

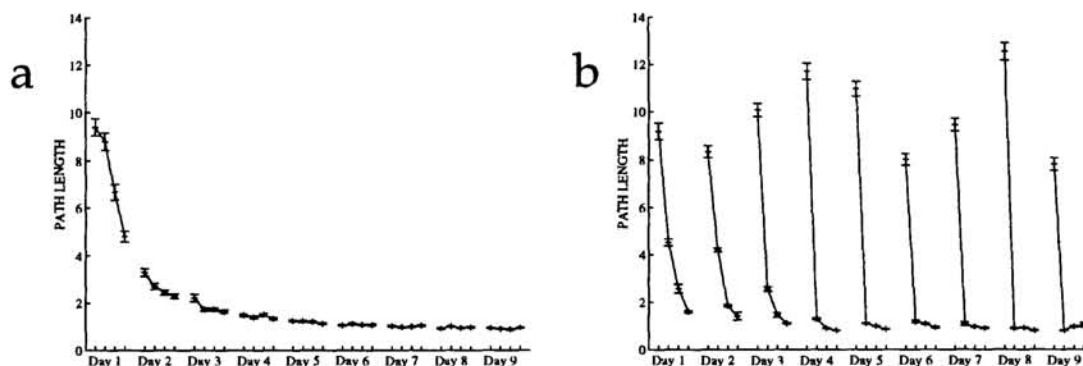

Figure 5: a.) Performance of the actor-critic model on the RMW task, and b.) performance of the full model on the DMP task. The data for comparison is shown in figures 1a and b.

the DMP task that the rats acquire over the course of the first few days of training. Interestingly, learning set is incomplete – on the first trial of each day, the rats still aim for the platform position on the previous day, even though this is never correct.[16] The significant differences in the path lengths on the first trial of each day (evidence in figure 1b and figure 5b) come from the relative placements of the platforms. However, the model did not use the same positions as the empirical data, and, in any case, the model of exploration behavior is rather simplistic.

The model demonstrates that reinforcement learning methods are perfectly fast enough to match empirical learning curves. This is fortunate, since, unlike most models specifically designed for open-field navigation,[6,4,5,20] RL methods can provably cope with substantially more complicated tasks with arbitrary barriers, *etc*, since they solve the temporal credit assignment problem in its full generality. The model also addresses the problem that coordinates in different parts of the same environment need to be mutually consistent, even if the animal only experiences some parts on separate trials. An important property of the model is that there is no requirement for the animal to have any explicit knowledge of the relationship between different place cells or place field position, size or shape. Such a requirement is imposed in various models.[9,4,6,20]

Experiments that are suggested by this model (as well as by certain others) concern the relationship between hippocampally dependent and independent spatial learning. First, once the coordinate system has been acquired, we predict that merely placing the rat at a new location would be enough to let it find the platform in one shot, though it might be necessary to reinforce the placement e.g. by first placing the rat in a bucket of cold water. Second, we know that the establishment of place fields in an environment happens substantially faster than establishment of one-shot or even ordinary learning to a platform.[23] We predict that blocking plasticity in the hippocampus *following* the establishment of place cells (possibly achieved without a platform) would *not* block learning of a platform. In fact, new experiments show that after extensive pre-training, rats can perform one-trial learning in the same environment to new platform positions on the DMP task without hippocampal synaptic plasticity.[16] This is in contrast to the effects of hippocampal lesion, which completely disrupts performance. According to the model, coordinates will have been learned during pre-training. The full prediction remains untested: that once place fields have been established, coordinates could be learned in the absence of hippocampal synaptic plasticity. A third prediction follows from evidence that rats with restricted hippocampal lesions can learn the fixed platform

task, but much more slowly, based on a gradual "shaping" procedure.[22] In our model, they may also be able to learn coordinates. However, a lengthy training procedure could be required, and testing might be complicated if expressing the knowledge required the use of hippocampus dependent short-term memory for the last platform location.[16]

One way of expressing the contribution of the hippocampus in the model is to say that its function is to provide a behavioural state space for the solution of complex tasks. Hence the contribution of the hippocampus to navigation is to provide place cells whose firing properties remain consistent in a given environment. It follows that in different behavioural situations, hippocampal cells should provide a representation based on something other than locations — and, indeed, there is evidence for this.[8] With regard to the role of the hippocampus in spatial tasks, the model demonstrates that the hippocampus may be fundamentally necessary without embodying a map.

## Footnotes

*Crichton Street, Edinburgh EH8 9LE, United Kingdom. Funded by Edin. Univ. Holdsworth Scholarship, the McDonnell-Pew foundation and NSF grant IBN-9634339. Email: djf@cfn.ed.ac.uk

# References

[1] Barto, AG & Sutton, RS (1981) *Biol. Cyber.*, **43**:1-8.

[2] Barto, AG, Sutton, RS & Anderson, CW (1983) *IEEE Trans. on Systems, Man and Cybernetics* **13**:834-846.

[3] Barto, AG, Sutton, RS & Watkins, CJCH (1989) *Tech Report 89-95*, CAIS, Univ. Mass., Amherst, MA.

[4] Blum, KI & Abbott, LF (1996) *Neural Computation*, **8**:85-93.

[5] Brown, MA & Sharp, PE (1995) *Hippocampus* **5**:171-188.

[6] Burgess, N, Recce, M & O'Keefe, J (1994) *Neural Networks*, **7**:1065-1081.

[7] Dayan, P (1991) *NIPS 3*, RP Lippmann et al, eds., 464-470.

[8] Eichenbaum, HB (1996) *Curr. Opin. Neurobiol.*, **6**:187-195.

[9] Gerstner, W & Abbott, LF (1996) *J. Computational Neurosci.* **4**:79-94.

[10] McNaughton, BL et al (1996) *J. Exp. Biol.*, **199**:173-185.

[11] Morris, RGM et al (1982) *Nature*, **297**:681-683.

[12] O'Keefe, J & Dostrovsky, J (1971) *Brain Res.*, **34**(171).

[13] Olton, DS & Samuelson, RJ (1976) *J. Exp. Psych: A.B.P.*, **2**:97-116. Rudy, JW & Sutherland, RW (1995) *Hippocampus*, **5**:375-389.

[14] Schultz, W, Dayan, P & Montague, PR (1997) *Science*, **275**, 1593-1599.

[15] Singh, SP Reinforcement learning with a hierarchy of abstract models.

[16] Steele, RJ & Morris, RGM *in preparation*.

[17] Sutton, RS (1988) *Machine Learning*, **3**:9-44.

[18] Taube, JS (1995) *J. Neurosci.* **15**(1):70-86.

[19] Tsitsiklis, JN & Van Roy, B (1996) *Tech Report LIDS-P-2322*, M.I.T.

[20] Wan, HS, Touretzky, DS & Redish, AD (1993) *Proc. 1993 Connectionist Models Summer School*, Lawrence Erlbaum, 11-19.

[21] Watkins, CJCH (1989) PhD Thesis, Cambridge.

[22] Whishaw, IQ & Jarrard, LF (1996) *Hippocampus*

[23] Wilson, MA & McNaughton, BL (1993) *Science* **261**:1055-1058.
